# On the Use of Projection Pursuit Constraints for Training Neural Networks

Nathan Intrator[*]
Computer Science Department
Tel-Aviv University
Ramat-Aviv, 69978 ISRAEL
and
Institute for Brain and Neural Systems,
Brown University
nin@math.tau.ac.il

## Abstract

We present a novel classification and regression method that combines exploratory projection pursuit (unsupervised training) with projection pursuit regression (supervised training), to yield a new family of cost/complexity penalty terms. Some improved generalization properties are demonstrated on real world problems.

## 1 Introduction

Parameter estimation becomes difficult in high-dimensional spaces due to the increasing sparseness of the data. Therefore, when a low dimensional representation is embedded in the data, dimensionality reduction methods become useful. One such method – projection pursuit regression (Friedman and Stuetzle, 1981) (PPR) is capable of performing dimensionality reduction by composition, namely, it constructs an approximation to the desired response function using a composition of lower dimensional smooth functions. These functions depend on low dimensional projections through the data.

---
[*]Research was supported by the National Science Foundation, the Army Research Office, and the Office of Naval Research.

When the dimensionality of the problem is in the thousands, even projection pursuit methods are almost always over-parametrized, therefore, additional smoothing is needed for low variance estimation. Exploratory Projection Pursuit (Friedman and Tukey, 1974; Friedman, 1987) (EPP) may be useful for that. It searches in a high dimensional space for structure in the form of (semi) linear projections with constraints characterized by a projection index. The projection index may be considered as a universal prior for a large class of problems, or may be tailored to a specific problem based on prior knowledge.

In this paper, the general form of exploratory projection pursuit is formulated to be an additional constraint for projection pursuit regression. In particular, a hybrid combination of supervised and unsupervised artificial neural network (ANN) is described as a special case. In addition, a specific projection index that is particularly useful for classification (Intrator, 1990; Intrator and Cooper, 1992) is introduced in this context. A more detailed discussion appears in Intrator (1993).

## 2   Brief Description of Projection Pursuit Regression

Let $(X, Y)$ be a pair of random variables, $X \in R^d$, and $Y \in R$. The problem is to approximate the $d$ dimensional surface

$$f(x) = E[Y|X = x]$$

from $n$ observations $(x_1, y_1), \ldots, (x_n, y_n)$.

PPR tries to approximate a function $f$ by a sum of ridge functions (functions that are constant along lines)

$$f(x) \simeq \sum_{j=1}^{m} g_j(a_j^T x).$$

The fitting procedure alternates between an estimation of a direction $\hat{a}$ and an estimation of a smooth function $g$, such that at iteration $j$, the square average of the residuals

$$r_{ij}(x_i) = r_{ij-1} - \hat{g}_j(\hat{a}_j^T x_i)$$

is minimized. This process is initialized by setting $r_{i0} = y_i$. Usually, the initial values of $a_j$ are taken to be the first few principal components of the data.

Estimation of the ridge functions can be achieved by various nonparametric smoothing techniques such as locally linear functions (Friedman and Stuetzle, 1981), k-nearest neighbors (Hall, 1989b), splines or variable degree polynomials. The smoothness constraint imposed on $g$, implies that the actual projection pursuit is achieved by minimizing at iteration $j$, the sum

$$\sum_{i=1}^{n} r_{ij}^2(x_i) + C(g_j),$$

for some smoothness measure $C$.

Although PPR converges to the desired response function (Jones, 1987), the use of non-parametric function estimation is likely to lead to overfitting. Recent results (Hornik, 1991) suggest that a feed forward network architecture with a single

hidden layer and a rather general fixed activation function is a universal approximator. Therefore, the use of a non-parametric single ridge function estimation can be avoided. It is thus appropriate to concentrate on the estimation of good projections. In the next section we present a general framework of PPR architecture, and in section 4 we restrict it to a feed-forward architecture with sigmoidal hidden units.

# 3 Estimating The Projections Using Exploratory Projection Pursuit

Exploratory projection pursuit is based on seeking *interesting* projections of high dimensional data points (Kruskal, 1969; Switzer, 1970; Kruskal, 1972; Friedman and Tukey, 1974; Friedman, 1987; Jones and Sibson, 1987; Hall, 1988; Huber, 1985, for review). The notion of interesting projections is motivated by an observation that for most high-dimensional data clouds, most low-dimensional projections are approximately normal (Diaconis and Freedman, 1984). This finding suggests that the important information in the data is conveyed in those directions whose single dimensional projected distribution is far from Gaussian. Various projection indices (measures for the goodness of a projection) differ on the assumptions about the nature of deviation from normality, and in their computational efficiency. They can be considered as different priors motivated by specific assumptions on the underlying model.

To partially decouple the search for a projection vector from the search for a non-parametric ridge function, we propose to add a penalty term, which is based on a projection index, to the energy minimization associated with the estimation of the ridge functions and the projections. Specifically, let $\rho(a)$ be a projection index which is minimized for projections with a certain deviation from normality; At the j'th iteration, we minimize the sum

$$\sum_i r_j^2(x_i) + C(g_j) + \rho(a_j).$$

When a concurrent minimization over several projections/functions is practical, we get a penalty term of the form

$$B(\hat{f}) = \sum_j [C(g_j) + \rho(a_j)].$$

Since $C$ and $\rho$ may not be linear, the more general measure that does not assume a stepwise approach, but instead seeks $l$ projections and ridge functions concurrently, is given by

$$B(\hat{f}) = C(g_1, \ldots, g_l) + \rho(a_1, \ldots, a_l),$$

In practice, $\rho$ depends implicitly on the training data, (the empirical density) and is therefore replaced by its empirical measure $\hat{\rho}$.

## 3.1 Some Possible Measures

Some applicable projection indices are discussed in (Huber, 1985; Jones and Sibson, 1987; Friedman, 1987; Hall, 1989a; Intrator, 1990). Probably, all the possible

measures should emphasize some form of deviation from normality but the specific type may depend on the problem at hand. For example, a measure based on the Karhunen Loève expansion (Mougeot et al., 1991) may be useful for image compression with autoassociative networks, since in this case one is interested in minimizing the $L^2$ norm of the distance between the reconstructed image and the original one, and under mild conditions, the Karhunen Loève expansion gives the optimal solution.

A different type of prior knowledge is required for classification problems. The underlying assumption then is that the data is clustered (when projecting in the right directions) and that the classification may be achieved by some (nonlinear) mapping of these clusters. In such a case, the projection index should emphasize multi-modality as a specific deviation from normality. A projection index that emphasizes multimodalities in the projected distribution (without relying on the class labels) has recently been introduced (Intrator, 1990) and implemented efficiently using a variant of a biologically motivated unsupervised network (Intrator and Cooper, 1992). Its integration into a back-propagation classifier will be discussed below.

## 3.2    Adding EPP constraints to back-propagation network

One way of adding some prior knowledge into the architecture is by minimizing the effective number of parameters using weight sharing, in which a single weight is shared among many connections in the network (Waibel et al., 1989; Le Cun et al., 1989). An extension of this idea is the "soft weight sharing" which favors irregularities in the weight distribution in the form of multimodality (Nowlan and Hinton, 1992). This penalty improved generalization results obtained by weight elimination penalty. Both these methods make an explicit assumption about the structure of the weight space, but with no regard to the structure of the input space.

As described in the context of projection pursuit regression, a penalty term may be added to the energy functional minimized by error back propagation, for the purpose of measuring directly the goodness of the projections sought by the network. Since our main interest is in reducing overfitting for high dimensional problems, our underlying assumption is that the surface function to be estimated can be faithfully represented using a low dimensional composition of sigmoidal functions, namely, using a back-propagation network in which the number of hidden units is *much smaller* than the number of input units. Therefore, the penalty term may be added only to the hidden layer. The synaptic modification equations of the hidden units' weights become

$$
\begin{aligned}
\frac{\partial w_{ij}}{\partial t} = \ &-\epsilon \Big[ \frac{\partial \mathcal{E}(w,x)}{\partial w_{ij}} \\
&+\frac{\partial \rho(w_1,\ldots,w_n)}{\partial w_{ij}} \\
&+(\text{Contribution of cost/complexity terms}) \Big].
\end{aligned}
$$

An approach of this type has been used in image compression, with a penalty aimed at minimizing the entropy of the projected distribution (Bichsel and Seitz, 1989). This penalty certainly measures deviation from normality, since entropy is maximized for a Gaussian distribution.

# 4 Projection Index for Classification: The Unsupervised BCM Neuron

Intrator (1990) has recently shown that a variant of the Bienenstock, Cooper and Munro neuron (Bienenstock et al., 1982) performs exploratory projection pursuit using a projection index that measures multi-modality. This neuron version allows theoretical analysis of some visual deprivation experiments (Intrator and Cooper, 1992), and is in agreement with the vast experimental results on visual cortical plasticity (Clothiaux et al., 1991). A network implementation which can find several projections in parallel while retaining its computational efficiency, was found to be applicable for extracting features from very high dimensional vector spaces (Intrator and Gold, 1993; Intrator et al., 1991; Intrator, 1992)

The activity of neuron $k$ in the network is $c_k = \sum_i x_i w_{ik} + w_{0k}$. The *inhibited* activity and threshold of the $k$'th neuron is given by

$$\tilde{c}_k = \sigma(c_k - \eta \sum_{j \neq k} c_j), \qquad \tilde{\Theta}_m^k = E[\tilde{c}_k^2].$$

The threshold $\tilde{\Theta}_m^k$ is the point at which the modification function $\phi$ changes sign (see Intrator and Cooper, 1992 for further details). The function $\phi$ is given by

$$\phi(c, \Theta_m) = c(c - \Theta_m).$$

The risk (projection index) for a single neuron is given by

$$R(w_k) = -\{\frac{1}{3}E[\tilde{c}_k^3] - \frac{1}{4}E^2[\tilde{c}_k^2]\}.$$

The total risk is the sum of each local risk. The negative gradient of the risk that leads to the synaptic modification equations is given by

$$\frac{\partial w_{ij}}{\partial t} = E[\phi(\tilde{c}_j, \Theta_m{}^j)\sigma'(\tilde{c}_j)x_i - \eta \sum_{k \neq j} \phi(\tilde{c}_k, \tilde{\Theta}_m^k)\sigma'(\tilde{c}_k)x_i].$$

This last equation is an additional penalty to the energy minimization of the supervised network. Note that there is an interaction between adjacent neurons in the hidden layer. In practice, the stochastic version of the differential equation can be used as the learning rule.

# 5 Applications

We have applied this hybrid classification method to various speech and image recognition problems in high dimensional space. In one speech application we used voiceless stop consonants extracted from the TIMIT database as training tokens (Intrator and Tajchman, 1991). A detailed biologically motivated speech representation was produced by Lyon's cochlear model (Lyon, 1982; Slaney, 1988). This representation produced 5040 dimensions (84 channels × 60 time slices). In addition to an initial voiceless stop, each token contained a final vowel from the set [aa, ao, er, iy]. Classification of the voiceless stop consonants using a test set that included 7 vowels [uh, ih, eh, ae, ah, uw, ow] produced an average error of 18.8%

while on the same task classification using back-propagation network produced an average error of 20.9% (a significant difference, P < .0013). Additional experiments on vowel tokens appear in Tajchman and Intrator (1992).

Another application is in the area of face recognition from gray level pixels (Intrator et al., 1992). After aligning and normalizing the images, the input was set to 37 × 62 pixels (total of 2294 dimensions). The recognition performance was tested on a subset of the MIT Media Lab database of face images made available by Turk and Pentland (1991) which contained 27 face images of each of 16 different persons. The images were taken under varying illumination and camera location. Of the 27 images available, 17 randomly chosen ones served for training and the remaining 10 were used for testing. Using an ensemble average of hybrid networks (Lincoln and Skrzypek, 1990; Pearlmutter and Rosenfeld, 1991; Perrone and Cooper, 1992) we obtained an error rate of 0.62% as opposed to 1.2% using a similar ensemble of back-prop networks. A single back-prop network achieves an error between 2.5% to 6% on this data. The experiments were done using 8 hidden units.

## 6    Summary

A penalty that allows the incorporation of additional prior information on the underlying model was presented. This prior was introduced in the context of projection pursuit regression, classification, and in the context of back-propagation network. It achieves partial decoupling of estimation of the ridge functions (in PPR) or the regression function in back-propagation net from the estimation of the projections. Thus it is potentially useful in reducing problems associated with overfitting which are more pronounced in high dimensional data.

Some possible projection indices were discussed and a specific projection index that is particularly useful for classification was presented in this context. This measure that emphasizes multi-modality in the projected distribution, was found useful in several very high dimensional problems.

### 6.1    Acknowledgments

I wish to thank Leon Cooper, Stu Geman and Michael Perrone for many fruitful conversations and to the referee for helpful comments. The speech experiments were performed using the computational facilities of the Cognitive Science Department at Brown University. Research was supported by the National Science Foundation, the Army Research Office, and the Office of Naval Research.

## References

Bichsel, M. and Seitz, P. (1989). Minimum class entropy: A maximum information approach to layered netowrks. *Neural Networks*, 2:133–141.

Bienenstock, E. L., Cooper, L. N., and Munro, P. W. (1982). Theory for the development of neuron selectivity: orientation specificity and binocular interaction in visual cortex. *Journal Neuroscience*, 2:32–48.

Clothiaux, E. E., Cooper, L. N., and Bear, M. F. (1991). Synaptic plasticity in visual cortex: Comparison of theory with experiment. *Journal of Neurophysiology*, 66:1785–1804.

Diaconis, P. and Freedman, D. (1984). Asymptotics of graphical projection pursuit. *Annals of Statistics*, 12:793–815.

Friedman, J. H. (1987). Exploratory projection pursuit. *Journal of the American Statistical Association*, 82:249–266.

Friedman, J. H. and Stuetzle, W. (1981). Projection pursuit regression. *Journal of the American Statistical Association*, 76:817–823.

Friedman, J. H. and Tukey, J. W. (1974). A projection pursuit algorithm for exploratory data analysis. *IEEE Transactions on Computers*, C(23):881–889.

Hall, P. (1988). Estimating the direction in which data set is most interesting. *Probab. Theory Rel. Fields*, 80:51–78.

Hall, P. (1989a). On polynomial-based projection indices for exploratory projection pursuit. *The Annals of Statistics*, 17:589–605.

Hall, P. (1989b). On projection pursuit regression. *The Annals of Statistics*, 17:573–588.

Hornik, K. (1991). Approximation capabilities of multilayer feedforward networks. *Neural Networks*, 4:251–257.

Huber, P. J. (1985). Projection pursuit. (with discussion). *The Annals of Statistics*, 13:435–475.

Intrator, N. (1990). Feature extraction using an unsupervised neural network. In Touretzky, D. S., Ellman, J. L., Sejnowski, T. J., and Hinton, G. E., editors, *Proceedings of the 1990 Connectionist Models Summer School*, pages 310–318. Morgan Kaufmann, San Mateo, CA.

Intrator, N. (1992). Feature extraction using an unsupervised neural network. *Neural Computation*, 4:98–107.

Intrator, N. (1993). Combining exploratory projection pursuit and projection pursuit regression with application to neural networks. *Neural Computation*. In press.

Intrator, N. and Cooper, L. N. (1992). Objective function formulation of the BCM theory of visual cortical plasticity: Statistical connections, stability conditions. *Neural Networks*, 5:3–17.

Intrator, N. and Gold, J. I. (1993). Three-dimensional object recognition of gray level images: The usefulness of distinguishing features. *Neural Computation*. In press.

Intrator, N., Gold, J. I., Bülthoff, H. H., and Edelman, S. (1991). Three-dimensional object recognition using an unsupervised neural network: Understanding the distinguishing features. In Feldman, Y. and Bruckstein, A., editors, *Proceedings of the 8th Israeli Conference on AICV*, pages 113–123. Elsevier.

Intrator, N., Reisfeld, D., and Yeshurun, Y. (1992). Face recognition using a hybrid supervised/unsupervised neural network. Preprint.

Intrator, N. and Tajchman, G. (1991). Supervised and unsupervised feature extraction from a cochlear model for speech recognition. In Juang, B. H., Kung, S. Y., and Kamm, C. A., editors, *Neural Networks for Signal Processing – Proceedings of the 1991 IEEE Workshop*, pages 460–469. IEEE Press, New York, NY.

Jones, L. (1987). On a conjecture of huber concerning the convergence of projection pursuit regression. *Annals of Statistics*, 15:880–882.

Jones, M. C. and Sibson, R. (1987). What is projection pursuit? (with discussion). *J. Roy. Statist. Soc.*, Ser. A(150):1–36.

Kruskal, J. B. (1969). Toward a practical method which helps uncover the structure of the set of multivariate observations by finding the linear transformation which optimizes a new 'index of condensation'. In Milton, R. C. and Nelder, J. A., editors, *Statistical Computation*, pages 427–440. Academic Press, New York.

Kruskal, J. B. (1972). Linear transformation of multivariate data to reveal clustering. In Shepard, R. N., Romney, A. K., and Nerlove, S. B., editors, *Multidimensional Scaling: Theory and Application in the Behavioral Sciences, I, Theory*, pages 179–191. Seminar Press, New York and London.

Le Cun, Y., Boser, B., Denker, J., Henderson, D., Howard, R., Hubbard, W., and Jackel, L. (1989). Backpropagation applied to handwritten zip code recognition. *Neural Computation*, 1:541–551.

Lincoln, W. P. and Skrzypek, J. (1990). Synergy of clustering multiple back-propagation networks. In Touretzky, D. S. and Lippmann, R. P., editors, *Advances in Neural Information Processing Systems*, volume 2, pages 650–657. Morgan Kaufmann, San Mateo, CA.

Lyon, R. F. (1982). A computational model of filtering, detection, and compression in the cochlea. In *Proceedings IEEE International Conference on Acoustics, Speech, and Signal Processing*, Paris, France.

Mougeot, M., Azencott, R., and Angeniol, B. (1991). Image compression with back propagation: Improvement of the visual restoration using different cost functions. *Neural Networks*, 4:467–476.

Nowlan, S. J. and Hinton, G. E. (1992). Simplifying neural networks by soft weight-sharing. *Neural Computation*. In press.

Pearlmutter, B. A. and Rosenfeld, R. (1991). Chaitin-kolmogorov complexity and generalization in neural networks. In Lippmann, R. P., Moody, J. E., and Touretzky, D. S., editors, *Advances in Neural Information Processing Systems*, volume 3, pages 925–931. Morgan Kaufmann, San Mateo, CA.

Perrone, M. P. and Cooper, L. N. (1992). When networks disagree: Generalized ensemble method for neural networks. In Mammone, R. J. and Zeevi, Y., editors, *Neural Networks: Theory and Applications*, volume 2. Academic Press.

Slaney, M. (1988). Lyon's cochlear model. Technical report, Apple Corporate Library, Cupertino, CA 95014.

Switzer, P. (1970). Numerical classification. In Barnett, V., editor, *Geostatistics*. Plenum Press, New York.

Tajchman, G. N. and Intrator, N. (1992). Phonetic classification of TIMIT segments preprocessed with lyon's cochlear model using a supervised/unsupervised hybrid neural network. In *Proceedings International Conference on Spoken Language Processing*, Banff, Alberta, Canada.

Turk, M. and Pentland, A. (1991). Eigenfaces for recognition. *J. of Cognitive Neuroscience*, 3:71–86.

Waibel, A., Hanazawa, T., Hinton, G., Shikano, K., and Lang, K. (1989). Phoneme recognition using time-delay neural networks. *IEEE Transactions on ASSP*, 37:328–339.
